# Use of a Multi-Layer Perceptron to Predict Malignancy in Ovarian Tumors

**Herman Verrelst,**
**Yves Moreau and Joos Vandewalle**
Dept. of Electrical Engineering
Katholieke Universiteit Leuven
Kard. Mercierlaan 94
B-3000 Leuven, Belgium

**Dirk Timmerman**

Dept. of Obst. and Gynaec.
University Hospitals Leuven
Herestraat 49
B-3000 Leuven, Belgium

## Abstract

We discuss the development of a Multi-Layer Perceptron neural network classifier for use in preoperative differentiation between benign and malignant ovarian tumors. As the Mean Squared classification Error is not sufficient to make correct and objective assessments about the performance of the neural classifier, the concepts of sensitivity and specificity are introduced and combined in Receiver Operating Characteristic curves. Based on objective observations such as sonomorphologic criteria, color Doppler imaging and results from serum tumor markers, the neural network is able to make reliable predictions with a discriminating performance comparable to that of experienced gynecologists.

## 1 Introduction

A reliable test for preoperative discrimination between benign and malignant ovarian tumors would be of considerable help to clinicians. It would assist them to select patients for whom minimally invasive surgery or conservative management suffices versus those for whom urgent referral to a gynecologic oncologist is needed.

We discuss the development of a neural network classifier/diagnostic tool. The neural network was trained by supervised learning, based on data from 191 thoroughly examined patients presenting with ovarian tumors of which 140 were benign and 51 malignant. As inputs to the network we chose indicators that in recent studies have proven their high predictive value [1, 2, 3]. Moreover, we gave preference to those indicators that can be obtained in an objective way by any gynecologist. Some of these indicators have already been used in attempts to make one single protocol or decision algorithm [3, 4].

In order to make reliable assessments on the practical performance of the classifier, it is necessary to work with other concepts than Mean Squared classification Error (MSE), which is traditionally used as a measure of goodness in the training of a neural network. We will introduce notions as specificity and sensitivity and combine them into Receiver Operating Characteristic (ROC) curves. The use of ROC-curves is motivated by the fact that they are independent of the relative proportion of the various output classes in the sample population. This enables an objective validation of the performance of the classifier. We will also show how, in the training of the neural network, MSE optimization with gradient methods can be refined and/or replaced with the help of ROC-curves and simulated annealing techniques.

The paper is organized as follows. In Section 2 we give a brief description of the selected input features. In Section 3 we state some drawbacks to the MSE criterion and introduce the concepts of sensitivity, specificity and ROC-curves. Section 4 then deals with the technicalities of training the neural network. In Section 5 we show the results and compare them to human performance.

## 2   Data acquisition and feature selection

The data were derived from a study group of 191 consecutive patients who were referred to a single institution (University Hospitals Leuven, Belgium) from August 1994 to August 1996. Table 1 lists the different indicators which were considered, together with their mean value and standard deviations or together with the relative presence in cases of benign and malignant tumors.

| Table 1 | Indicator | Benign | Malignant |
|---|---|---|---|
| *Demographic* | Age | $49.3 \pm 16.0$ | $58.3 \pm 14.3$ |
| | Postmenopausal | 40% | 70.6% |
| *Serum marker* | CA 125 (log) | $2.8 \pm 1.1$ | $5.2 \pm 1.9$ |
| *CDI* | Blood flow present | 72.9% | 100% |
| *Morphologic* | Abdominal fluid | 12.1% | 52.9% |
| | Bilateral mass | 11.4% | 35.3% |
| | Unilocular cyst | 42.1% | 5.9% |
| | Multiloc/solid cyst | 16.4% | 49.0% |
| | Smooth wall | 58.6% | 2.0% |
| | Irregular wall | 32.1% | 76.5% |
| | Papillations | 7.9% | 74.5% |

Table 1: Demographic, serum marker, color Doppler imaging and morphologic indicators. For the continuous valued features the mean and standard deviation for each class are reported. For binary valued indicators, the last two columns give the presence of the feature in both classes e.g. only 2% of malignant tumors had smooth walls.

First, all patients were scanned with ultrasonography to obtain detailed gray-scale images of the tumors. Every tumor was extensively examined for its morphologic characteristics. Table 1 lists the selected morphologic features: presence of abdominal fluid collection, papillary structures ($> 3mm$), smooth internal walls, wall irregularities, whether the cysts were unilocular, multilocular-solid and/or present on both pelvic sides. All outcomes are binary valued: every observation relates to the presence (1) or absence (0) of these characteristics.

Secondly, all tumors were entirely surveyed by color Doppler imaging to detect presence or absence of blood flow within the septa, cyst walls, solid tumor areas or ovarian tissue. The outcome is also binary valued (1/0).

Thirdly, in 173 out of the total of 191 patients, serum CA 125 levels were measured, using CA 125 II immunoradiometric assays (Centocor, Malvern, PA). The CA 125 antigen is a glycoprotein that is expressed by most epithelial ovarian cancers. The numerical value gives the concentration in U/ml. Because almost all values were situated in a small interval between 0 and 100, and because a small portion took values up to 30,000, this variable was rescaled by taking its logarithm.

Since age and menopausal status of the patient are considered to be highly relevant, these are also included. The menopausal score is $-1$ for premenopausal, $+1$ for postmenopausal. A third class of patients were assigned a 0 value. These patients had had an hysterectomy, so no menopausal status could be appointed to them.

It is beyond the scope of this paper to give a complete account of the meaning of the different features that are used or the way in which the data were acquired. We will limit ourselves to this short description and refer the reader to [2, 3] and gynecological textbooks for a more detailed explanation.

## 3 Receiver Operating Characteristics

### 3.1 Drawbacks to Mean Squared classification Error

Let us assume that we use a one-hidden-layer feed-forward NN with $m$ inputs $x_k^i$, $n_h$ hidden neurons with the tanh(.) as activation function, and one output $\hat{y}_k$,

$$y_k(\theta) = \sum_{j=1}^{n_h} w_j \tanh(\sum_{i=1}^{m} v_{ij} x_k^i + \beta_j), \tag{1}$$

parameterized by the vector $\theta$ consisting of the network's weights $w_j$ and $v_{ij}$ and bias terms $\beta_j$. The cost function is often chosen to be the squared difference between the desired $d_k$ and the actual response $y_k$, averaged over all $N$ samples [12],

$$J(\theta) = \frac{1}{N} \sum_{k=1}^{N} (d_k - y_k(\theta))^2. \tag{2}$$

This type of cost function is continuous and differentiable, so it can be used in gradient based optimization techniques such as steepest descent (back-propagation), quasi-Newton or Levenberg-Marquardt methods [8, 9, 11, 12]. However there are some drawbacks to the use of this type of cost function.

**First of all, the MSE is heavily dependent on the relative proportion of the different output classes in the training set.** In our dichotomic case this can easily be demonstrated by writing the cost function, with superscripts $b$ and $m$ respectively meaning benign and malignant, as

$$J(\theta) = \underbrace{\frac{N_b}{N_b + N_m}}_{\lambda} \frac{1}{N_b} \sum_{k=1}^{N_b} (d_k^b - y_k)^2 + \underbrace{\frac{N_m}{N_b + N_m}}_{(1-\lambda)} \frac{1}{N_m} \sum_{k=1}^{N_m} (d_k^m - y_k)^2 \tag{3}$$

If the relative proportion in the sample population is not representative for reality, the $\lambda$ parameter should be adjusted accordingly. In practice this real proportion is often not known accurately or one simply ignores the meaning of $\lambda$ and uses it as a design parameter in order to bias the accuracy towards one of the output classes.

**A second drawback of the MSE cost function is that it is not very informative towards practical usage of the classification tool.** A clinician is not interested in the averaged deviation from desired numbers, but thinks in terms of percentages found, missed or misclassified. In the next section we will introduce the concepts of sensitivity and specificity to express these more practical measures.

## 3.2 Sensitivity, specificity and ROC-curves

If we take the desired response to be 0 for benign and 1 for malignant cases, the way to make clearcut (dichotomic) decisions is to compare the numerical outcome of the neural network to a certain threshold value $T$ between 0 and 1. When the outcome is above the threshold $T$, the prediction is said to be *positive*. Otherwise the prediction is said to be *negative*. With this convention, we say that the prediction was

| | |
|---|---|
| **True Positive (TP)** | if the prediction was positive when the sample was malignant. |
| **True Negative (TN)** | if the prediction was negative when the sample was benign. |
| **False Positive (FP)** | if the prediction was positive when the sample was benign. |
| **False Negative (FN)** | if the prediction was negative when the sample was malignant. |

To every of the just defined terms $TP$, $TN$, $FP$ and $FN$, a certain subregion of the total sample space can be associated, as depicted in Figure 1. In the same sense, we can associate to them a certain number counting the samples in each subregion. We can then define sensitivity as $\frac{TP}{TP+FN}$, the proportion of malignant cases that

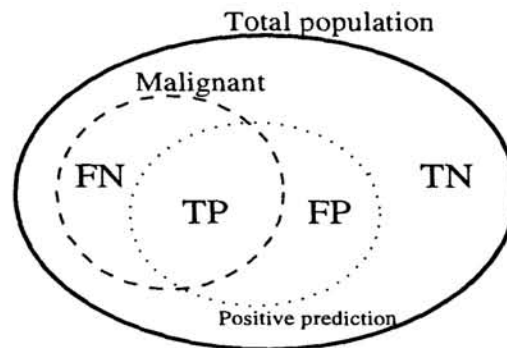

Figure 1: The concepts of true and false positive and negative illustrated. The dashed area indicates the malignant cases in the total sample population. The positive prediction of an imperfect classification (dotted area) does not fully coincide with this sub area.

are predicted to be malignant and specificity as $\frac{TN}{FP+TN}$, the proportion of benign cases that are predicted to be benign. The false positive rate is 1−specificity.

When varying the threshold $T$, the values of $TP$, $TN$, $FP$, $FN$ and therefore also sensitivity and specificity, will change. A low threshold will detect almost all malignant cases at the cost of many false positives. A high threshold will give less false positives, but will also detect less malignant cases. Receiver Operating Characteristic (ROC) curves are a way to visualize this relationship. The plot gives the sensitivity versus false positive rate for varying thresholds $T$ (e.g. Figure 2).

The ROC-curve is useful and widely used device for assessing and comparing the value of tests [5, 7]. The proportion of the whole area of the graph which lies below the ROC-curve is a one-value measure of the accuracy of a test [6]. The higher this proportion, the better the test. Figure 2 shows the ROC-curves for two simple classifiers that use only one single indicator. (Which means that we classify a tumor being malignant when the value of the indicator rises above a certain value.) It is seen that the CA 125 level has high predictive power as its ROC-curve spans 87.5% of the total area (left Figure 2). For the age parameter, the ROC-curve spans only 65.6% (right Figure 2). As indicated by the horizontal line in the plot, a CA 125 level classification will only misclassify 15% of all benign cases to reach a 80% sensitivity, whereas using only age, one would then misclassify up to 50% of them.

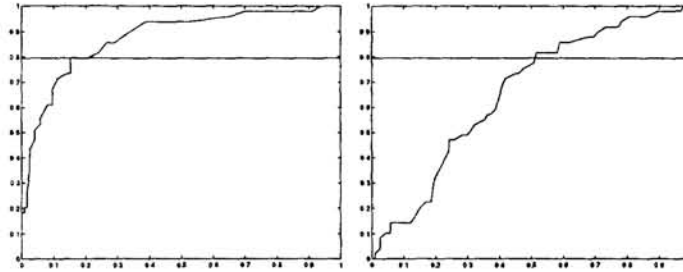

Figure 2: The Receiver Operating Characteristic (ROC) curve is the plot of the sensitivity versus the false positive rate of a classifier for varying thresholds used. Only single indicators (left: CA 125, right: age) are used for these ROC-curves. The horizontal line marks the 80% specificity level.

Since for every set of parameters of the neural network the area under the ROC-curve can be calculated numerically, this one-value measure can also be used for supervised training, as will be shown in the next Section.

## 4  Simulation results

### 4.1  Inputs and architecture

The continuous inputs were standardized by subtracting their mean and dividing by their standard deviation (both calculated over the entire population). Binary valued inputs were left unchanged. The desired outputs were labeled 0 for benign examples, 1 for malignant cases. The data set was split up: 2/3 of both benign and malignant samples were randomly selected to form the training set. The remaining examples formed the test set. The ratio of benign to all examples is $\lambda \approx \frac{2}{3}$.

Since the training set is not large, there is a risk of overtraining when too many parameters are used. We will limit the number of hidden neurons to $n_h = 3$ or 5. As the CA 125 level measurement is more expensive and time consuming, we will investigate two different classifiers: one which does use the CA 125 level and one which does not. The one-hidden-layer MLP architectures that are used, are 11-3-1 and 10-5-1. A tanh(.) is taken for the activation function in the hidden layer.

### 4.2  Training

A first way of training was MSE optimization using the cost function (3). By taking $\lambda = \frac{1}{3}$ in this expression, the role of malignant examples is more heavily weighted. The parameter vector $\theta$ was randomly initialized (zero mean Gaussian distribution, standard deviation $\sigma = 0.01$). Training was done using a quasi-Newton method with BFGS-update of the Hessian ($fminu$ in Matlab) [8, 9]. To prevent overtraining, the training was stopped before the MSE on the test set started to rise. Only few iterations ($\approx 100$) were needed.

A second way of training was through the use of the area spanned by the ROC-curve of the classifier and simulated annealing techniques [10]. The area-measure $A^{ROC}$ was numerically calculated for every set of trial parameters: first the sensitivity and false positive rate were calculated for 1000 increasing values of the threshold $T$ between 0 and 1, which gave the ROC-curve; secondly the area $A^{ROC}$ under the curve was numerically calculated with the trapezoidal integration rule.

We used Boltzmann Simulated Annealing to maximize the ROC-area. At time $k$ a trial parameter set of the neural network $\theta_{k+1}$ is randomly generated in the neighborhood of the present set $\theta_k$ (Gaussian distribution, $\sigma = 0.001$). The trial set $\theta_{k+1}$ is always accepted if the area $A^{ROC}_{k+1} \geq A^{ROC}_k$. If $A^{ROC}_{k+1} < A^{ROC}_k$, $\theta_{k+1}$ is accepted if

$$e^{\left(\frac{A^{ROC}_{k+1} - A^{ROC}_k}{A^{ROC}_k}\right)/T_e} > \alpha$$

with $\alpha$ a uniformly distributed random variable $\in [0,1]$ and $T_e$ the temperature. As cooling schedule we took

$$T_e = 1/(100 + 10k),$$

so that the annealing was low-temperature and fast-cooling. The optimization was stopped before the ROC-area calculated for the test set started to decrease. Only a few hundred annealing epochs were allowed.

## 4.3 Results

Table 2 states the results for the different approaches. One can see that adding the CA 125 serum level clearly improves the classifier's performance. Without it, the ROC-curve spans about 96.5% of the total square area of the plot, whereas with the CA 125 indicator it spans almost 98%. Also, the two training methods are seen to give comparable results. Figure 3 shows the ROC-curve calculated for the total population for the 11-3-1 MLP case, trained with simulated annealing

| Table 2 | Training set | Test set | Total population |
|---|---|---|---|
| 10-5-1 MLP, MSE | 96.7% | 96.4% | 96.5% |
| 10-5-1 MLP, SA | 96.6% | 96.2% | 96.4% |
| 11-3-1 MLP, MSE | 97.9% | 97.4% | 97.7% |
| 11-3-1 MLP, SA | 97.9% | 97.5% | 97.8% |

Table 2: For the two architectures (10-5-1 and 11-3-1) of the MLP and for the gradient (MSE) and the simulated annealing (SA) optimization techniques, this table gives the resulting areas under the ROC-curves.

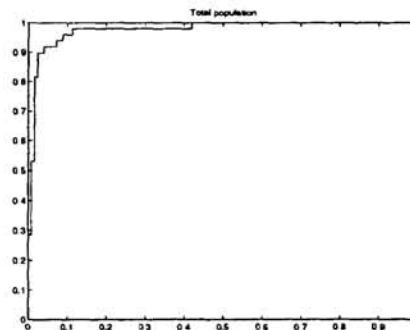

Figure 3: ROC-curves of 11-3-1 MLP (with CA 125 level indicator), trained with simulated annealing. The curve, calculated for the total population, spans 97.8% of the total region.

All patients were examined by two gynecologists, who gave their subjective impressions and also classified the ovarian tumors into (probably) benign and malignant. Histopathological examinations of the tumors afterwards showed these gynecologists

to have a sensitivity up to 98% and a false positive rate of 13% and 12% respectively. As can be seen in Figure 3, the 11-3-1 MLP has a similar performance. For a sensitivity of 98%, its false positive rate is between 10% and 15%.

## 5   Conclusion

In this paper we have discussed the development of a Multi-Layer Perceptron neural network classifier for use in preoperative differentiation between benign and malignant ovarian tumors. To assess the performance and for training the classifiers, the concepts of sensitivity and specificity were introduced and combined in Receiver Operating Characteristic curves. Based on objective observations available to every gynecologist, the neural network is able to make reliable predictions with a discriminating performance comparable to that of experienced gynecologists.

### Acknowledgments

This research work was carried out at the ESAT laboratory and the Interdisciplinary Center of Neural Networks ICNN of the Katholieke Universiteit Leuven, in the following frameworks: the Belgian Programme on Interuniversity Poles of Attraction, initiated by the Belgian State, Prime Minister's Office for Science, Technology and Culture (IUAP P4-02 and IUAP P4-24), a Concerted Action Project MIPS (Modelbased Information Processing Systems) of the Flemish Community and the FWO (Fund for Scientific Research - Flanders) project G.0262.97 : Learning and Optimization: an Interdisciplinary Approach. The scientific responsibility rests with its authors.

## References

[1] Bast R. C., Jr., Klug T.L. St. John E., et al, "A radioimmunoassay using a monoclonal antibody to monitor the course of epithelial ovarian cancer," *N. Engl. J. Med.*, Vol. 309, pp. 883-888, 1983

[2] Timmerman D., Bourne T., Tailor A., Van Assche F.A., Vergote I., "Preoperative differentiation between benign and malignant adnexal masses," *submitted*

[3] Tailor A., Jurkovic D., Bourne T.H., Collins W.P., Campbell S., "Sonographic prediction of malignancy in adnexal masses using multivariate logistic regression analysis," *Ultrasound Obstet. Gynaecol.* in press, 1997

[4] Jacobs I., Oram D., Fairbanks J., et al., "A risk of malignancy index incorporating CA 125, ultrasound and menopausal status for the accurate preoperative diagnosis of ovarian cancer," *Br. J. Obstet. Gynaecol.*, Vol. 97, pp. 922-929, 1990

[5] Hanley J.A., McNeil B., "A method of comparing the areas under the receiver operating characteristics curves derived from the same cases," *Radiology*, Vol. 148, pp. 839-843, 1983

[6] Swets J.A., "Measuring the accuracy of diagnostic systems," *Science*, Vol. 240, pp. 1285-1293, 1988

[7] Galen R.S., Gambino S., *Beyond normality: the predictive value and efficiency of medical diagnosis*, John Wiley, New York, 1975.

[8] Gill P., Murray W., Wright M., *Practical Optimization*, Acad. Press, New York, 1981

[9] Fletcher R., *Practical methods of optimization*, 2nd ed., John Wiley, New York, 1987.

[10] Kirkpatrick S., Gelatt C.D., Vecchi M., "Optimization by simulated annealing," *Science*, Vol. 220, pp. 621-680, 1983.

[11] Rumelhart D.E., Hinton G.E., Williams R.J., "Learning representations by back-propagating errors," *Nature*, Vol. 323, pp. 533-536, 1986.

[12] Bishop C., *Artificial Neural Networks for Pattern Recognition*, OUP, Oxford, 1996